# Analysis of Spectral Kernel Design based Semi-supervised Learning

**Tong Zhang**
Yahoo! Inc.
New York City, NY 10011

**Rie Kubota Ando**
IBM T. J. Watson Research Center
Yorktown Heights, NY 10598

## Abstract

We consider a framework for semi-supervised learning using spectral decomposition based un-supervised kernel design. This approach subsumes a class of previously proposed semi-supervised learning methods on data graphs. We examine various theoretical properties of such methods. In particular, we derive a generalization performance bound, and obtain the optimal kernel design by minimizing the bound. Based on the theoretical analysis, we are able to demonstrate why spectral kernel design based methods can often improve the predictive performance. Experiments are used to illustrate the main consequences of our analysis.

## 1 Introduction

Spectral graph methods have been used both in clustering and in semi-supervised learning. This paper focuses on semi-supervised learning, where a classifier is constructed from both labeled and unlabeled training examples. Although previous studies showed that this class of methods work well for certain concrete problems (for example, see [1, 4, 5, 6]), there is no satisfactory theory demonstrating why (and under what circumstances) such methods should work.

The purpose of this paper is to develop a more complete theoretical understanding for graph based semi-supervised learning. In Theorem 2.1, we present a transductive formulation of kernel learning on graphs which is equivalent to supervised kernel learning. This new kernel learning formulation includes some of the previous proposed graph semi-supervised learning methods as special cases. A consequence is that we can view such graph-based semi-supervised learning methods as kernel design methods that utilize unlabeled data; the designed kernel is then used in the standard supervised learning setting. This insight allows us to prove useful results concerning the behavior of graph based semi-supervised learning from the more general view of spectral kernel design. Similar spectral kernel design ideas also appeared in [2]. However, they didn't present a graph-based learning formulation (Theorem 2.1 in this paper); nor did they study the theoretical properties of such methods. We focus on two issues for graph kernel learning formulations based on Theorem 2.1. First, we establish the convergence of graph based semi-supervised learning (when the number of unlabeled data increases). Second, we obtain a learning bound, which can be used to compare the performance of different kernels. This analysis gives insights to what are good kernels, and why graph-based spectral kernel design is often helpful in various applications. Examples are given to justify the theoretical analysis. Due to the space limitations, proofs

will not be included in this paper.

## 2 Transductive Kernel Learning on Graphs

We shall start with notations for supervised learning. Consider the problem of predicting a real-valued output $Y$ based on its corresponding input vector $X$. In the standard machine learning formulation, we assume that the data $(X, Y)$ are drawn from an unknown underlying distribution $D$. Our goal is to find a predictor $p(x)$ so that the expected true loss of $p$ given below is as small as possible: $R(p(\cdot)) = E_{(X,Y) \sim D} L(p(X), Y)$, where we use $E_{(X,Y) \sim D}$ to denote the expectation with respect to the true (but unknown) underlying distribution $D$. Typically, one needs to restrict the hypothesis function family size so that a stable estimate within the function family can be obtained from a finite number of samples. We are interested in learning in Hilbert spaces. For notational simplicity, we assume that there is a feature representation $\psi(x) \in \mathcal{H}$, where $\mathcal{H}$ is a high (possibly infinity) dimensional feature space. We denote $\psi(x)$ by column vectors, so that the inner product in the Hilbert-space $\mathcal{H}$ is the vector product. A linear classifier $p(x)$ on $\mathcal{H}$ can be represented by a vector $w \in \mathcal{H}$ such that $p(x) = w^T \psi(x)$.

Let the training samples be $(X_1, Y_1), \ldots, (X_n, Y_n)$. We consider the following regularized linear prediction method on $\mathcal{H}$:

$$\hat{p}(x) = \hat{w}^T \psi(x), \quad \hat{w} = \arg\min_{w \in \mathcal{H}} \left[ \frac{1}{n} \sum_{i=1}^{n} L(w^T \psi(X_i), Y_i) + \lambda w^T w \right]. \quad (1)$$

If $\mathcal{H}$ is an infinite dimensional space, then it is not be feasible to solve (1) directly. A remedy is to use kernel methods. Given a feature representation $\psi(x)$, we can define kernel $k(x, x') = \psi(x)^T \psi(x')$. It is well-known (the so-called representer theorem) that the solution of (1) can be represented as $\hat{p}(x) = \sum_{i=1}^{n} \hat{\alpha}_i k(X_i, x)$, where $[\hat{\alpha}_i]$ is given by

$$[\hat{\alpha}_i] = \arg\min_{[\alpha_i] \in R^n} \left[ \frac{1}{n} \sum_{i=1}^{n} L \left( \sum_{j=1}^{n} \alpha_j k(X_i, X_j), Y_i \right) + \lambda \sum_{i,j=1}^{n} \alpha_i \alpha_j k(X_i, X_j) \right]. \quad (2)$$

The above formulations of kernel methods are standard. In the following, we present an equivalence of supervised kernel learning to a specific semi-supervised formulation. Although this representation is implicit in some earlier papers, the explicit form of this method is not well-known. As we shall see later, this new kernel learning formulation is critical for analyzing a class of graph-based semi-supervised learning methods.

In this framework, the *data graph* consists of nodes that are the data points $X_j$. The edge connecting two nodes $X_i$ and $X_j$ is weighted by $k(X_i, X_j)$. The following theorem, which establishes the graph kernel learning formulation we will study in this paper, essentially implies that graph-based semi-supervised learning is equivalent to the supervised learning method which employs the same kernel.

**Theorem 2.1 (Graph Kernel Learning)** *Consider labeled data $\{(X_i, Y_i)\}_{i=1,\ldots,n}$ and unlabeled data $X_j$ ($j = n+1, \ldots, m$). Consider real-valued vectors $f = [f_1, \ldots, f_m]^T \in R^m$, and the following semi-supervised learning method:*

$$\hat{f} = \arg\inf_{f \in R^m} \left[ \frac{1}{n} \sum_{i=1}^{n} L(f_i, Y_i) + \lambda f^T K^{-1} f \right], \quad (3)$$

*where $K$ (often called gram-matrix in kernel learning or affinity matrix in graph learning) is an $m \times m$ matrix with $K_{i,j} = k(X_i, X_j) = \psi(X_i)^T \psi(X_j)$. Let $\hat{p}$ be the solution of (1), then $\hat{f}_j = \hat{p}(X_j)$ for $j = 1, \ldots, m$.*

The kernel gram matrix $K$ is always positive semi-definite. However, if $K$ is not full rank (singular), then the correct interpretation of $f^T K^{-1} f$ is $\lim_{\mu \to 0^+} f^T (K + \mu I_{m \times m})^{-1} f$, where $I_{m \times m}$ is the $m \times m$ identity matrix. If we start with a given kernel $k$ and let $K = [k(X_i, X_j)]$, then a semi-supervised learning method of the form (3) is equivalent to the supervised method (1). It follows that with a formulation like (3), the only way to utilize unlabeled data is to replace $K$ by a kernel $\bar{K}$ in (3), or $k$ by $\bar{k}$ in (2), where $\bar{K}$ (or $\bar{k}$) depends on the unlabeled data. In other words, the only benefit of unlabeled data in this setting is to construct a good kernel based on unlabeled data.

Some of previous graph-based semi-supervised learning methods employ the same formulation (3) with $K^{-1}$ replaced by the graph Laplacian operator $L$ (which we will describe in Section 5). However, the equivalence of this formulation and supervised kernel learning (with kernel matrix $K = L^{-1}$) was not obtained in these earlier studies. This equivalence is important for good theoretical understanding, as we will see later in this paper. Moreover, by treating graph-based supervised learning as unsupervised kernel design (see Figure 1), the scope of this paper is more general than graph Laplacian based methods.

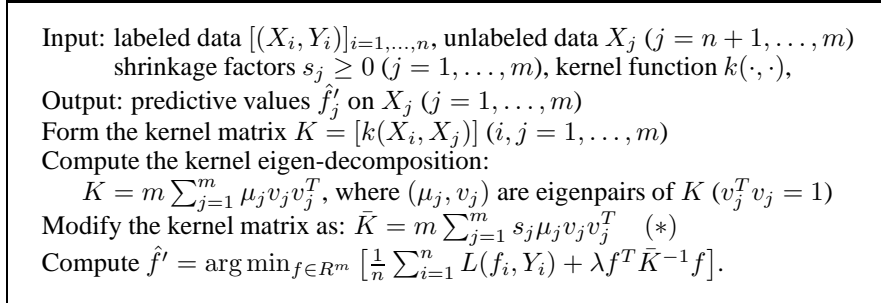

Figure 1: Spectral kernel design based semi-supervised learning on graph

In Figure 1, we consider a general formulation of semi-supervised learning method on data graph through spectral kernel design. This is the method we will analyze in the paper. As a special case, we can let $s_j = g(\mu_j)$ in Figure 1, where $g$ is a rational function, then $\bar{K} = g(K/m)K$. In this special case, we do not have to compute eigen-decomposition of $K$. Therefore we obtain a simpler algorithm with the $(*)$ in Figure 1 replaced by

$$\bar{K} = g(K/m)K. \tag{4}$$

As mentioned earlier, the idea of using spectral kernel design has appeared in [2] although they didn't base their method on the graph formulation (3). However, we believe our analysis also sheds lights to their methods. The semi-supervised learning method described in Figure 1 is useful only when $\hat{f}'$ is a better predictor than $\hat{f}$ in Theorem 2.1 (which uses the original kernel $K$) – in other words, only when the new kernel $\bar{K}$ is better than $K$.

In the next few sections, we will investigate the following issues concerning the theoretical behavior of this algorithm: (a) the limiting behavior of $\hat{f}'$ as $m \to \infty$; that is, whether $\hat{f}'_j$ converges for each $j$; (b) the generalization performance of (3); (c) optimal Kernel design by minimizing the generalization error, and its implications; (d) statistical models under which spectral kernel design based semi-supervised learning is effective.

## 3 The Limiting Behavior of Graph-based Semi-supervised Learning

We want to show that as $m \to \infty$, the semi-supervised algorithm in Figure 1 is well-behaved. That is, $\hat{f}'_j$ converges as $m \to \infty$. This is one of the most fundamental issues.

Using feature space representation, we have $k(x, x') = \psi(x)^T \psi(x')$. Therefore a change of kernel can be regarded as a change of feature mapping. In particular, we consider a feature transformation of the form $\bar{\psi}(x) = S^{1/2}\psi(x)$, where $S$ is an appropriate positive semi-definite operator on $\mathcal{H}$. The following result establishes an equivalent feature space formulation of the semi-supervised learning method in Figure 1.

**Theorem 3.1** *Using notations in Figure 1. Assume $k(x, x') = \psi(x)^T \psi(x')$. Consider $S = \sum_{j=1}^m s_j u_j u_j^T$, where $u_j = \Psi v_j/\sqrt{\mu_j}$, $\Psi = [\psi(X_1), \ldots, \psi(X_m)]$, then $(\mu_j, u_j)$ is an eigenpair of $\Psi\Psi^T/m$. Let*

$$\hat{p}'(x) = \hat{w}'^T S^{1/2}\psi(x), \qquad \hat{w}' = \arg\min_{w \in \mathcal{H}} \left[ \frac{1}{n} \sum_{i=1}^n L(w^T S^{1/2}\psi(X_i), Y_i) + \lambda w^T w \right].$$

*Then $\hat{f}'_j = \hat{p}'(X_j)$ $(j = 1, \ldots, m)$.*

The asymptotic behavior of Figure 1 when $m \to \infty$ can be easily understood from Theorem 3.1. In this case, we just replace $\Psi\Psi^T/m = \frac{1}{m}\sum_{j=1}^m \psi(X_j)\psi(X_j)^T$ by $\mathbf{E}_X \psi(X)\psi(X)^T$. The spectral decomposition of $\mathbf{E}_X \psi(X)\psi(X)^T$ corresponds to the feature space PCA. It is clear that if $S$ converges, then the feature space algorithm in Theorem 3.1 also converges. In general, $S$ converges if the eigenvectors $u_j$ converges and the shrinkage factors $s_j$ are bounded. As a special case, we have the following result.

**Theorem 3.2** *Consider a sequence of data $X_1, X_2, \ldots$ drawn from a distribution, with only the first $n$ points labeled. Assume when $m \to \infty$, $\sum_{j=1}^m \psi(X_j)\psi(X_j)^T/m$ converges to $\mathbf{E}_X \psi(X)\psi(X)^T$ almost surely, and $g$ is a continuous function in the spectral range of $\mathbf{E}_X \psi(X)\psi(X)^T$. Now in Figure 1 with $(*)$ given by (4) and kernel $k(x, x') = \psi(x)^T \psi(x')$, $\hat{f}'_j$ converges almost surely for each fixed $j$.*

# 4 Generalization analysis on graph

We study the generalization behavior of graph based semi-supervised learning algorithm (3), and use it to compare different kernels. We will then use this bound to justify the kernel design method given in Section 2. To measure the sample complexity, we consider $m$ points $(X_j, Y_j)$ for $i = 1, \ldots, m$. We randomly pick $n$ distinct integers $i_1, \ldots, i_n$ from $\{1, \ldots, m\}$ uniformly (sample without replacement), and regard it as the $n$ labeled training data. We obtain predictive values $\hat{f}_j$ on the graph using the semi-supervised learning method (3) with the labeled data, and test it on the remaining $m - n$ data points. We are interested in the average predictive performance over all random draws.

**Theorem 4.1** *Consider $(X_j, Y_j)$ for $i = 1, \ldots, m$. Assume that we randomly pick $n$ distinct integers $i_1, \ldots, i_n$ from $\{1, \ldots, m\}$ uniformly (sample without replacement), and denote it by $Z_n$. Let $\hat{f}(Z_n)$ be the semi-supervised learning method (3) using training data in $Z_n$: $\hat{f}(Z_n) = \arg\min_{f \in R^m} \left[ \frac{1}{n} \sum_{i \in Z_n} L(f_i, Y_i) + \lambda f^T K^{-1} f \right]$. If $|\frac{\partial}{\partial p} L(p, y)| \leq \gamma$, and $L(p, y)$ is convex with respect to $p$, then we have*

$$\mathbf{E}_{Z_n} \frac{1}{m - n} \sum_{j \notin Z_n} L(\hat{f}_j(Z_n), Y_j) \leq \inf_{f \in R^m} \left[ \frac{1}{m} \sum_{j=1}^m L(f_j, Y_j) + \lambda f^T K^{-1} f + \frac{\gamma^2 \text{tr}(K)}{2\lambda nm} \right].$$

The bound depends on the regularization parameter $\lambda$ in addition to the kernel $K$. In order to compare different kernels, it is reasonable to compare the bound with the optimal $\lambda$ for

each $K$. That is, in addition to minimizing $f$, we also minimize over $\lambda$ on the right hand of the bound. Note that in practice, it is usually not difficult to find a nearly-optimal $\lambda$ through cross validation, implying that it is reasonable to assume that we can choose the optimal $\lambda$ in the bound. With the optimal $\lambda$, we obtain:

$$\mathbf{E}_{Z_n} \frac{1}{m-n} \sum_{j \notin Z_n} L(\hat{f}_j(Z_n), Y_j) \leq \inf_{f \in R^m} \left[ \frac{1}{m} \sum_{j=1}^{m} L(f_j, Y_j) + \frac{\gamma}{\sqrt{2n}} \sqrt{R(f, K)} \right],$$

where $R(f, K) = \operatorname{tr}(K/m) f^T K^{-1} f$ is the complexity of $f$ with respect to kernel $K$.

If we define $\bar{K}$ as in Figure 1, then the complexity of a function $f$ with respect to $\bar{K}$ is given by $R(f, \bar{K}) = (\sum_{j=1}^{m} s_j \mu_j)(\sum_{j=1}^{m} \alpha_j^2 / (s_j \mu_j))$. If we believe that a good approximate target function $f$ can be expressed as $f = \sum_j \alpha_j v_j$ with $|\alpha_j| \leq \beta_j$ for some known $\beta_j$, then based on this belief, the optimal choice of the shrinkage factor becomes $s_j = \beta_j / \mu_j$. That is, the kernel that optimizes the bound is $\bar{K} = \sum_j \beta_j v_j v_j^T$, where $v_j$ are normalized eigenvectors of $K$. In this case, we have $R(f, \bar{K}) \leq (\sum_j \beta_j)^2$. The eigenvalues of the optimal kernel is thus independent of $K$, but depends only on the spectral coefficient's range $\beta_j$ of the approximate target function.

Since there is no reason to believe that the eigenvalues $\mu_j$ of the original kernel $K$ are proportional to the target spectral coefficient range. If we have some guess of the spectral coefficients of the target, then one may use the knowledge to obtain a better kernel. This justifies why spectral kernel design based algorithm can be potentially helpful (when we have some information on the target spectral coefficients). In practice, it is usually difficult to have a precise guess of $\beta_j$. However, for many application problems, we observe in practice that the eigenvalues of kernel $K$ decays more slowly than that of the target spectral coefficients. In this case, our analysis implies that we should use an alternative kernel with faster eigenvalue decay: for example, using $K^2$ instead of $K$. This has a dimension reduction effect. That is, we effectively project the data into the principal components of data. The intuition is also quite clear: if the dimension of the target function is small (spectral coefficient decays fast), then we should project data to those dimensions by reducing the remaining noisy dimensions (corresponding to fast kernel eigenvalue decay).

## 5    Spectral analysis: the effect of input noise

We provide a justification on why spectral coefficients of the target function often decay faster than the eigenvalues of a natural kernel $K$. In essence, this is due to the fact that input vector $X$ is often corrupted with noise. Together with results in the previous section, we know that in order to achieve optimal performance, we need to use a kernel with faster eigenvalue decay. We will demonstrate this phenomenon under a statistical model, and use the feature space notation in Section 3. For simplicity, we assume that $\psi(x) = x$.

We consider a two-class classification problem in $R^\infty$ (with the standard 2-norm inner-product), where the label $Y = \pm 1$. We first start with a noise free model, where the data can be partitioned into $p$ clusters. Each cluster $\ell$ is composed of a single center point $\bar{x}_\ell$ (having zero variance) with label $\bar{y}_\ell = \pm 1$. In this model, assume that the centers are well separated so that there is a weight vector $w_*$ such that $w_*^T w_* < \infty$ and $w_*^T \bar{x}_\ell = \bar{y}_\ell$. Without loss of generality, we may assume that $\bar{x}_\ell$ and $w_*$ belong to a $p$-dimensional subspace $V_p$. Let $V_p^\perp$ be its orthogonal complement. Assume now that the observed input data are corrupted with noise. We first generate a center index $\ell$, and then noise $\delta$ (which may depend on $\ell$). The observed input data is the corrupted data $X = \bar{x}_\ell + \delta$, and the observed output is $Y = w_*^T \bar{x}_\ell$. In this model, let $\ell(X_i)$ be the center corresponding to $X_i$, the observation can be decomposed as: $X_i = \bar{x}_{\ell(X_i)} + \delta(X_i)$, and $Y_i = w_*^T \bar{x}_{\ell(X_i)}$. Given noise $\delta$, we

decompose it as $\delta = \delta_1 + \delta_2$ where $\delta_1$ is the orthogonal projection of $\delta$ in $V_p$, and $\delta_2$ is the orthogonal projection of $\delta$ in $V_p^{\perp}$. We assume that $\delta_1$ is a small noise component; the component $\delta_2$ can be large but has small variance in every direction.

**Theorem 5.1** *Consider the data generation model in this section, with observation $X = \bar{x}_\ell + \delta$ and $Y = w_*^T \bar{x}_\ell$. Assume that $\delta$ is conditionally zero-mean given $\ell$: $\mathbf{E}_{\delta|\ell}\delta = 0$. Let $\mathbf{E}XX^T = \sum_j \mu_j u_j u_j^T$ be the spectral decomposition with decreasing eigenvalues $\mu_j$ ($u_j^T u_j = 1$). Then the following claims are valid: let $\sigma_1^2 \geq \sigma_2^2 \geq \cdots$ be the eigenvalues of $\mathbf{E}\delta_2\delta_2^T$, then $\mu_j \geq \sigma_j^2$; if $\|\delta_1\|_2 \leq b/\|w_*\|_2$, then $|w_*^T X_i - Y_i| \leq b$; $\forall t \geq 0$, $\sum_{j \geq 1}(w_*^T u_j)^2 \mu_j^{-t} \leq w_*^T (\mathbf{E}\,\bar{x}_\ell \bar{x}_\ell^T)^{-t} w_*$.*

Consider $m$ points $X_1, \ldots, X_m$. Let $\Psi = [X_1, \ldots, X_m]$ and $K = \Psi^T \Psi = m \sum_j \mu_j v_j v_j^T$ be the kernel spectral decomposition. Let $u_j = \Psi v_j / \sqrt{m\mu_j}$, $f_i = w_*^T X_i$, and $f = \sum_j \alpha_j v_j$. Then it is not difficult to verify that $\alpha_j = \sqrt{m\mu_j} w_*^T u_j$. If we assume that asymptotically $\frac{1}{m}\sum_{i=1}^m X_i X_i^T \to \mathbf{E}XX^T$, then we have the following consequences:

- $f_i = w_*^T X_i$ is a good approximate target when $b$ is small. In particular, if $b < 1$, then this function always gives the correct class label.
- For all $t > 0$, the spectral coefficient $\alpha_j$ of $f$ decays as $\frac{1}{m}\sum_{j=1}^m \alpha_j^2 / \mu_j^{1+t} \leq w_*^T (\mathbf{E}\bar{x}_\ell \bar{x}_\ell^T)^{-t} w_*$.
- The eigenvalue $\mu_j$ decays slowly when the noise spectral decays slowly: $\mu_j \geq \sigma_j^2$.

If the clean data are well behaved in that we can find a weight vector such that $w_*^T (\mathbf{E}_X \bar{x}_{\ell(X)} \bar{x}_{\ell(X)}^T)^{-t} w_*$ is bounded for some $t > 1$, then when the data are corrupted with noise, we can find a good approximate target that has spectral decay faster (on average) than that of the kernel eigenvalues. This analysis implies that if the feature representation associated with the original kernel is corrupted with noise, then it is often helpful to use a kernel with faster spectral decay. For example, instead of using $K$, we may use $\bar{K} = K^2$. However, it may not be easy to estimate the exact decay rate of the target spectral coefficients. In practice, one may use cross validation to optimize the kernel.

A kernel with fast spectral decay projects the data into the most prominent principal components. Therefore we are interested in designing kernels which can achieve a dimension reduction effect. Although one may use direct eigenvalue computation, an alternative is to use a function $g(K/m)K$ for such an effect, as in (4). For example, we may consider a normalized kernel such that $K/m = \sum_j \mu_j u_j u_j^T$ where $0 \leq u_j \leq 1$. A standard normalization method is to use $D^{-1/2}KD^{-1/2}$, where $D$ is the diagonal matrix with each entry corresponding to the row sums of $K$. It follows that $g(K/m)K = m\sum_j g(\mu_j)\mu_j u_j u_j^T$. We are interested in a function $g$ such that $g(\mu)\mu \approx 1$ when $\mu \in [\alpha, 1]$ for some $\alpha$, and $g(\mu)\mu \approx 0$ when $\mu < \alpha$ (where $\alpha$ is close to 1). One such function is to let $g(\mu)\mu = (1-\alpha)/(1-\alpha\mu)$. This is the function used in various graph Laplacian formulations with normalized Gaussian kernel as the initial kernel $K$. For example, see [5]. Our analysis suggests that it is the dimension reduction effect of this function that is important, rather than the connection to graph Laplacian. As we shall see in the empirical examples, other kernels such as $K^2$, which achieve similar dimension reduction effect (but has nothing to do with graph Laplacian), also improve performance.

# 6   Empirical Examples

This section shows empirical examples to demonstrate some consequences of our theoretical analysis. We use the MNIST data set (http://yann.lecun.com/exdb/mnist/), consisting

of hand-written digit images (representing 10 classes, from digit "0" to digit "9"). In the following experiments, we randomly draw $m = 2000$ samples. We regard $n = 100$ of them as labeled data, and the remaining $m - n = 1900$ as unlabeled test data.

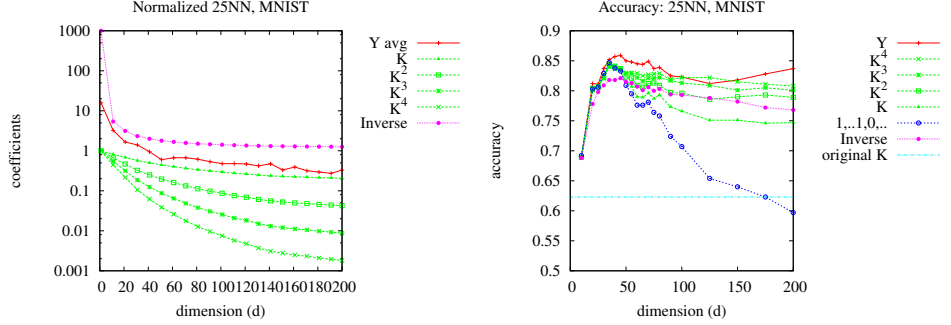

Figure 2: Left: spectral coefficients; right: classification accuracy.

Throughout the experiments, we use the least squares loss: $L(p, y) = (p - y)^2$ for simplicity. We study the performance of various kernel design methods, by changing the spectral coefficients of the initial gram matrix $K$, as in Figure 1. Below we write $\bar{\mu}_j$ for the new spectral coefficient of the new gram matrix $\bar{K}$: i.e., $\bar{K} = \sum_{i=1}^{m} \bar{\mu}_i v_i v_i^T$. We study the following kernel design methods (also see [2]), with a dimension cut off parameter $d$, so that $\bar{\mu}_i = 0$ when $i > d$. (a) $[1, \ldots, 1, 0, \ldots, 0]$: $\bar{\mu}_i = 1$ if $i \leq d$, and 0 otherwise. This was used in spectral clustering [3]. (b) $K$: $\bar{\mu}_i = \mu_i$ if $i \leq d$; 0 otherwise. This method is essentially kernel principal component analysis which keeps the $d$ most significant principal components of $K$. (c) $K^p$: $\bar{\mu}_i = \mu_i^p$ if $i \leq d$; 0 otherwise. We set $p = 2, 3, 4$. This accelerates the decay of eigenvalues of $K$. (d) Inverse: $\bar{\mu}_i = 1/(1 - \rho \mu_i)$ if $i \leq d$; 0 otherwise. $\rho$ is a constant close to 1 (we used 0.999). This is essentially graph-Laplacian based semi-supervised learning for normalized kernel (e.g. see [5]). Note that the standard graph-Laplacian formulation sets $d = m$. (e) $Y$: $\bar{\mu}_i = |Y^T v_i|$ if $i \leq d$; 0 otherwise. This is the oracle kernel that optimizes our generalization bound. The purpose of testing this oracle method is to validate our analysis by checking whether good kernel in our theory produces good classification performance on real data. Note that in the experiments, we use averaged $Y$ over the ten classes. Therefore the resulting kernel will not be the best possible kernel for each specific class, and thus its performance may not always be optimal.

Figure 2 shows the spectral coefficients of the above mentioned kernel design methods and the corresponding classification performance. The initial kernel is normalized 25-NN, which is defined as $K = D^{-1/2} W D^{-1/2}$ (see previous section), where $W_{ij} = 1$ if either the $i$-th example is one of the 25 nearest neighbors of the $j$-th example or vice versa; and 0 otherwise. As expected, the results demonstrate that the target spectral coefficients $Y$ decay faster than that of the original kernel $K$. Therefore it is useful to use kernel design methods that accelerate the eigenvalue decay. The accuracy plot on the right is consistent with our theory. The near oracle kernel 'Y' performs well especially when the dimension cut-off is large. With appropriate dimension $d$, all methods perform better than the supervised base-line (original K) which is below 65%. With appropriate dimension cut-off, all methods perform similarly (over 80%). However, $K^p$ with ($p = 2, 3, 4$) is less sensitive to the cut-off dimension $d$ than the kernel principal component dimension reduction method $K$. Moreover, the hard threshold method in spectral clustering ($[1, \ldots, 1, 0, \ldots, 0]$) is not stable. Similar behavior can also be observed with other initial kernels. Figure 3 shows the classification accuracy with the standard Gaussian kernel as the initial kernel $K$, both with and without normalization. We also used different bandwidth $t$ to illustrate that the

behavior of different methods are similar with different $t$ (in a reasonable range). The result shows that normalization is not critical for achieving high performance, at least for this data. Again, we observe that the near oracle method performs extremely well. The spectral clustering kernel is sensitive to the cut-off dimension, while $K^p$ with $p = 2, 3, 4$ are quite stable. The standard kernel principal component dimension reduction (method $K$) performs very well with appropriately chosen dimension cut-off. The experiments are consistent with our theoretical analysis.

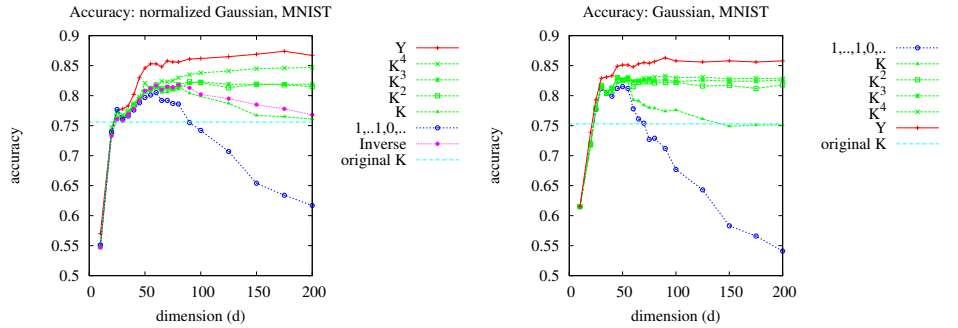

Figure 3: Classification accuracy with Gaussian kernel $k(i, j) = \exp(-||x_i - x_j||_2^2/t)$. Left: normalized Gaussian ($t = 0.1$); right: unnormalized Gaussian ($t = 0.3$).

## 7 Conclusion

We investigated a class of graph-based semi-supervised learning methods. By establishing a graph-based formulation of kernel learning, we showed that this class of semi-supervised learning methods is equivalent to supervised kernel learning with unsupervised kernel design (explored in [2]). We then obtained a generalization bound, which implies that the eigenvalues of the optimal kernel should decay at the same rate of the target spectral coefficients. Moreover, we showed that input noise can cause the target spectral coefficients to decay faster than the kernel spectral coefficients. The analysis explains why it is often helpful to modify the original kernel eigenvalues to achieve a dimension reduction effect.

## References

[1] Mikhail Belkin and Partha Niyogi. Semi-supervised learning on Riemannian manifolds. *Machine Learning*, Special Issue on Clustering:209–239, 2004.

[2] Olivier Chapelle, Jason Weston, and Bernhard Sch:olkopf. Cluster kernels for semi-supervised learning. In *NIPS*, 2003.

[3] Andrew Y. Ng, Michael I. Jordan, and Yair Weiss. On spectral clustering: Analysis and an algorithm. In *NIPS*, pages 849–856, 2001.

[4] M. Szummer and T. Jaakkola. Partially labeled classification with Markov random walks. In *NIPS 2001*, 2002.

[5] D. Zhou, O. Bousquet, T.N. Lal, J. Weston, and B. Schlkopf. Learning with local and global consistency. In *NIPS 2003*, pages 321–328, 2004.

[6] Xiaojin Zhu, Zoubin Ghahramani, and John Lafferty. Semi-supervised learning using Gaussian fields and harmonic functions. In *ICML 2003*, 2003.
